# Morton-Style Factorial Coding of Color in Primary Visual Cortex

**Javier R. Movellan**
Institute for Neural Computation
University of California San Diego
La Jolla, CA 92093-0515
*movellan@inc.ucsd.edu*

**Thomas Wachtler**
Sloan Center for Theoretical Neurobiology
The Salk Institute
La Jolla, CA 92037, USA
*thomas@salk.edu*

**Thomas D. Albright**
Howard Hughes Medical Institute
The Salk Institute
La Jolla, CA 92037, USA
*tom@salk.edu*

**Terrence Sejnowski**
Computational Neurobiology Laboratory
The Salk Institute
La Jolla, CA 92037, USA
*terry@salk.edu*

## Abstract

We introduce the notion of Morton-style factorial coding and illustrate how it may help understand information integration and perceptual coding in the brain. We show that by focusing on average responses one may miss the existence of factorial coding mechanisms that become only apparent when analyzing spike count histograms. We show evidence suggesting that the classical/non-classical receptive field organization in the cortex effectively enforces the development of Morton-style factorial codes. This may provide some cues to help understand perceptual coding in the brain and to develop new unsupervised learning algorithms. While methods like ICA (Bell & Sejnowski, 1997) develop independent codes, in Morton-style coding the goal is to make two or more external aspects of the world become independent when conditioning on internal representations.

In this paper we introduce the notion of Morton-style factorial coding and illustrate how it may help analyze information integration and perceptual organization in the brain. In the neurosciences factorial codes are often studied in the context of mean tuning curves. A tuning curve is called separable if it can be expressed as the product of terms selectively influenced by different stimulus dimensions. Separable tuning curves are taken as evidence of factorial coding mechanisms. In this paper we show that by focusing on average responses one may miss the existence of factorial coding mechanisms that become only apparent when analyzing spike count histograms.

Morton (1969) analyzed a wide variety of psychophysical experiments on word perception and showed that they could be explained using a model in which stimulus and context have separable effects on perception. More precisely, in Mortons' model the joint effect of stimulus and context on a perceptual representation can be obtained by multiplying terms

selectively controlled by stimulus and by context, i.e.,

$$P(r \mid s, c) = \frac{\eta_s(s,r)\eta_c(c,r)}{\sum_r \eta_s(s,r)\eta_c(c,r)} \qquad (1)$$

where $P(r \mid s, c)$ is the empirical probability of perceiving the perceptual alternative $r$ in response to stimulus $s$ in context $c$, $\eta_s(s,r)$ represents the support of stimulus $s$ for percept $r$ and $\eta_c(c,r)$ the support of the context for percept $r$. Massaro (1987b, 1987a, 1989a) has shown that this form of factorization describes accurately a wide variety of psychophysical studies in domains such as word recognition, phoneme recognition, audiovisual speech recognition, and recognition of facial expressions.

Morton-style factorial codes used to be taken as evidence for a feedforward coding mechanism (Massaro, 1989b) but Movellan & McClelland (2001) showed that neural networks with feedback connections can develop factorial codes when they follow an architectural constraint named "channel separability". Channel separability is defined as follows: First we identify the neurons which have a direct influence on the observed responses (e.g., the set of neurons that affect an electrode). For a given set of response units, the stimulus chanel is defined as the set of units modulated by the stimulus provided the response specification units are excised from the rest of the network. The context channel is the set of units modulated by the context provided the response units are excised from the rest of the networks. Two channels are called separable if they have no units in common. Channel separability implies that the influences of an information source upon the channel of another information source should be mediated via the response specification units (see Figure 1). While the models used in Movellan and McClelland (2001) are a simplification of actual neural circuits, the analysis suggests that the form of separability expressed in the the Morton-Massaro model may be a useful paradigm for the study of information integration in the brain. Indeed it is quite remarkable that the functional organization of cortex into classical/non-classical receptive fields provides a separable architecture (See Figure 1). Such organization may be nature's way of enforcing Morton-style perceptual coding. In this paper we present evidence in favor of this view by investigating how color is encoded in primary visual cortex.

It is well known that stimuli of equal chromaticity can evoke different color percepts, depending on the visual context (Wesner & Shevell, 1992; Brown & MacLeod, 1997). Context dependent responses to color stimuli have been found in V4 (Zeki, 1983). More recently the last three authors of this article investigated the chromatic tuning properties of V1 cells in response to stimuli presented in different chromatic contexts (Wachtler, Sejnowski, & Albright, 2003). The experiment showed that the background color, outside the cell's classical receptive field, had a significant effect on the response to colors inside the receptive field. No attempt was made to model the form of such influence. In this paper we analyze quantitatively the results of that experiment and show that a large proportion of these neurons, adhered to the Morton-Massaro law, i.e., stimulus and context had a separable influence on the spike count histograms of these cells.

## 1 Methods

The animal preparation and methods of this experiment are described in Wachtler et al. (in press) in great detail. Here we briefly describe the portion of the experiment relevant to us. Two adult female rhesus monkeys were used in the study. Extracellular potentials from single isolated neurons were recorded from two macaque monkeys. The monkeys were awake and were required to fixate a small fixation target for the duration of each trial (2500 ms.). Amplified electrical activity from the cortex was passed to a data acquisition system for spike detection and sorting. Once a neuron was isolated, its receptive field was determined using flashed and moving bars of different size, orientation, and color. All the

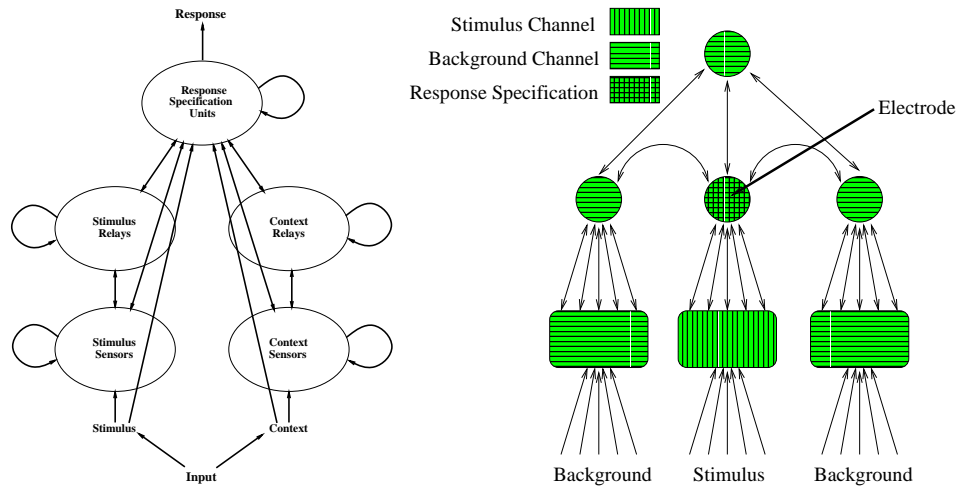

Figure 1: Left: A network with separable context and stimulus processing channels. Right: The arrows connecting the stimulus to the unit in the center represent the classical receptive field of that unit. External inputs affecting the classical receptive field are called "stimuli" and all the other inputs are called "background". In this preparation the stimulus and background channels are separable.

neurons recorded had receptive fields at eccentricities between $2°$ and $5°$.

Once the receptive fields were located, the color tuning of the neurons was mapped by flashing 8 stimuli of different chromaticity. The stimuli were homogenous color squares, centered on and at least twice as large as the receptive field of the neuron under study. They were flashed for 500 ms. Chromaticity was defined in a color space similar to the one used in Derrington, Krauskopf, and Lennie (1984). Cone excitations were calculated on the basis of the human cone fundamentals proposed by Stockman (Stockman, MaCleod, & Johnson, 1993). The origin of the color space corresponded to a homogeneous gray background to which the animal had been adapted (luminance 48 cd/m$^2$). The three coordinate axis of the color space corresponded to L versus M-cone contrast, S-cone contrast, and achromatic luminance. The 8 color stimuli were isoluminant with the gray background, had a fixed color contrast (distance from origin of color space) and had chromatic directions corresponding to polar angles $45°$.

After several presentations of the stimuli, the chromatic directions for which the neurons showed a clear response were determined, and one of them was selected as the second background condition. In the second condition, the color of the background changed during stimulus presentation (i.e., for 500 ms) to a different color. This color was isoluminant with the gray background, was in the direction of a stimulus color to which the cell showed clear response, but was of lower chromatic contrast than the stimulus colors. In subsequent trials combinations of the 8 stimulus and 2 background conditions were presented in random order.

For each trial we recored the number of spikes in a 100 ms window starting 50 ms after stimulus onset. This time window was chosen because color tuning was usually more pronounced in the first response phase as compared to later periods of the response and because it maximized the effects of context. Data were recorded for a total of 94 units. Of these, 20 neurons were selected for having the strongest background effect and a minimum of 16 trials per condition. No other criteria were used for the selection of these neurons.

## 2 Results

Figure 2 shows example tuning curves of 4 different neurons. The thick lines represent the average response for a particular color stimulus in the plane defined by the first two chromatic axis. The dark curve represents responses for the gray background condition. The light curve represents responses for the color background condition. The boxes around the tuning curves represent average response rates as a function of stimulus onset for the two background conditions.

Testing whether a code is factorial is like testing for the absence of interaction terms in Analysis of Variance (ANOVA). The complexity (i.e., degrees of freedom) of an ANOVA model without interaction terms is identical to the complexity of the Morton-Massaro model. When testing for interaction effects we analyze whether the addition of interaction terms provides significant improvement on data fit over a simple additive model. In our case we investigate whether the addition of non-factorial terms provides a significant improvement on data fit over the factorial Morton-Massaro model. For each neuron there were 8 stimulus conditions, 2 background conditions, and 10 response alternatives, one per bin in the spike count histogram. The probabilities of the spike count histogram add up to one thus, there is a total of $8 \times 2 \times 9 = 144$ independent probability estimates per neuron. In this case the Morton-Massaro model requires $(10 - 1) \times (8 + 2 - 1) = 81$ parameters (Movellan & McClelland, 2001), thus there is a total of 63 nonfactorial terms.

For each neuron we fitted Morton-Massaro's model and performed a standard likelihood test to see whether the additional nonfactorial terms improved data fit significantly (i.e., whether the deviations from the Morton-Massaro factorial model where significant). We found that of the 20 neurons only 5 showed significant deviations from the Morton-Massaro model (chi-square test, 63 degrees of freedom, $p < 0.05$). While the Morton-Massaro model had 81 parameters many of them were highly redundant. We also evaluated a 30 parameter version of the model by performing PCA independently on the stimulus and on the context parameters of the full model and deleting coefficients with small eigenvalues. The 30 parameter model provided fits almost indistinguishable from the 81 parameter model. In this case only 4 neurons showed significant deviations from the model (chi-square, 124 df, $p < 0.05$). On a pool of 20 neurons compliant with the Morton-Massaro model one would expect the test to mistakenly reject 1 neuron by chance. Rejection of 4 or more neurons out of 20 is not inconsistent with the idea that all the neurons were in fact compliant with the Morton-Massaro model ($p > 0.20$, binomial test).

Figure 2 shows the obtained and predicted spike count histograms for a typical neuron. The top row represents the 8 stimulus conditions with gray background. The bottom row shows the 8 conditions with color background. Lines represent spike count histograms predicted by the Morton-Massaro model, dots represent obtained spike count histograms.

In order to test the statistical power of the likelihood-ratio test, we generated 20 neurons with random histograms. The histograms were unimodal, with peak response randomly selected between 0 and 9, with fall-offs similar to those found in the actual neurons and with the same number of observations per condition as in the actual neurons. We then fitted the 81-parameter Morton-Massaro model to each of these neurons and tested it using a likelihood ratio test. All the simulated neurons exhibited statistically significant deviations from the model (chi-square, 63 df, $p < 0.05$) suggesting that the test was quite sensitive.

Finally, for comparison purposes we tested a model of information integration that uses the same number of parameters as the Morton-Massaro model but in which the stimulus and context terms are are combined additively instead of multiplicatively, i.e.,

$$P(r \mid s, c) = \frac{\eta_s(s, r) + \eta_c(c, r)}{\sum_r \eta_s(s, r) + \eta_c(c, r)} \tag{2}$$

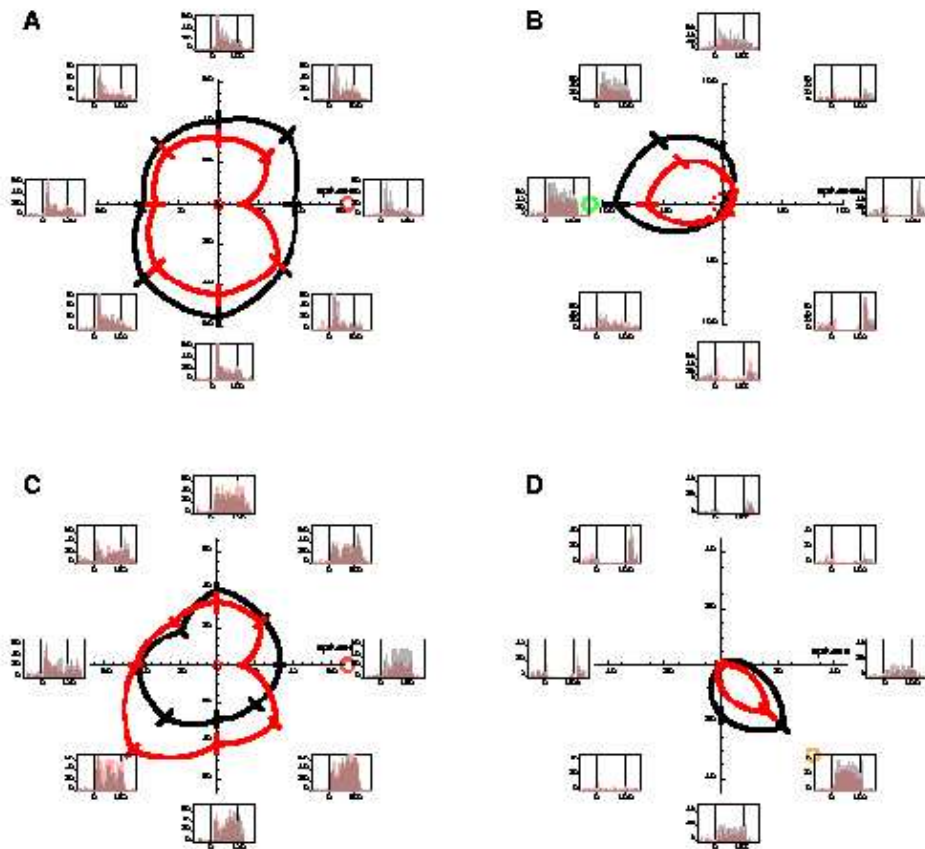

Figure 2: Effect of the stimulus and background on the chromatic mean tuning curves of 4 neurons. The thick dark and light lines show mean responses in the isoluminant plane (x axis: L-M cone variation; y axis: S cone variation) for the two background conditions. Black: gray background; Light: colored background. The 8 boxes around each tuning curve shows the average response rate as a function of the time from stimulus onset for the two background conditions.

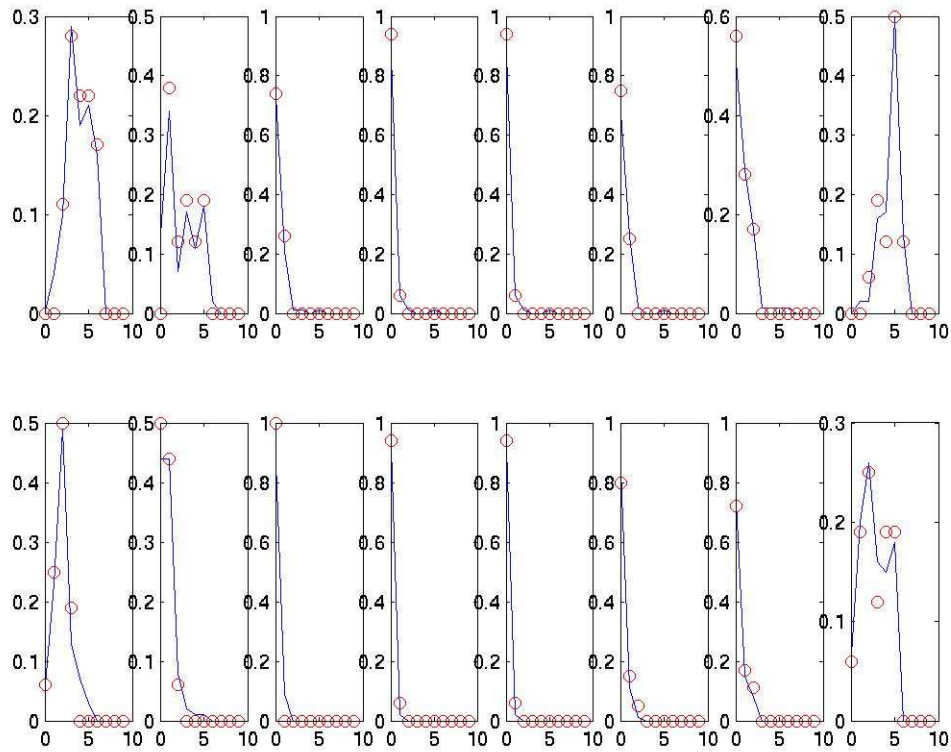

Figure 3: Predicted (lines) and obtained (dots) spike count histograms for a typical neuron. The horizontal axis represents spike counts in a 100 ms. window. The vertical axis represents probabilities. Each row represents a different background condition. Each column represents a different stimulus condition.

After fitting the new model, we performed a likelihood-ratio test. 80 % of the neurons showed significant deviations from this model (chi-square, 63 df, $p < 0.05$).

## 3   Relation to Tuning Curve Separability

In neuroscience separability is commonly studied in the context of mean tuning curves. For example, a tuning curve is called (multiplicatively) separable if the conditional expected value of a neuron's response can be decomposed as the product of two different factors each selectively influenced by a single stimulus dimension. An important aspect of the Morton-Massaro model is that it applies to entire response histograms, not to expected values. If the Morton-Massaro model holds, then separability appears in the following sense: If we are allowed to see the response histograms for all the stimuli in background condition A and the response histogram for a reference stimulus in background condition B, then it should be possible to predict the response histograms for any stimulus in background condition B. For example, by looking at the top row of Figure 1 and one of the cells of the bottom row of Figure 1, it should be possible to reproduce all the other cells in the bottom row.

Obviously if we can predict response histograms then we can also predict tuning curves, since they are based on averages of response histograms. Most importantly, there are forms of separability of the tuning curve that become only apparent when studying the entire response histogram. Figure 4 illustrates this fact with an example. The curve shows the tuning curves of a particular neuron from an experiment fitted using the Morton-Massaro model. These curves were obtained by fitting the entire spike count histograms for each stimulus and background condition, and then obtaining the mean response for the predicted histograms. The large open circles represent the obtained average responses. The dots represent 95 % confidence intervals around those responses. Note that the two tuning curves do not appear separable in a discernable way (it is not possible to predict curve B by looking at curve A and a single point of curve B). Separability becomes only apparent when the entire histogram is analyzed, not just the tuning curves based on response averages.

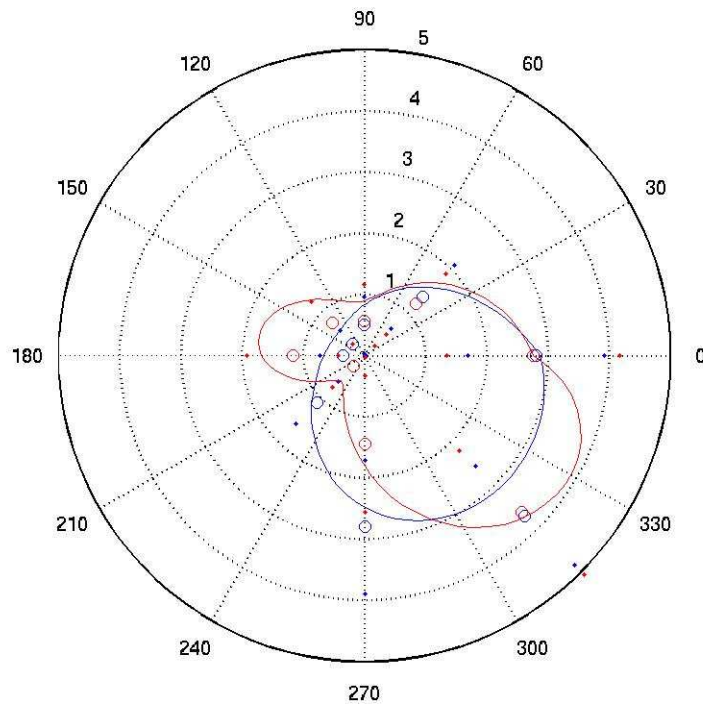

Figure 4: Tuning curves for a typical neuron as predicted by the Morton-Massaro model. The two curves represent the average response of the neuron to isoluminant stimulus, for two different background conditions. The elongated curve corresponds to the homogenous gray background and the circular curve to the colored background. The open dots are the obtained mean responses. The dots represent 95 % confidence interval of those responses. Note that the predicted curves do not appear separable in a classic sense. However since they are generated by Morton's model the underlying code is factorial. This becomes apparent only when one looks at spike count histograms, not just mean tuning curves.

# 4 Discussion

We introduced the notion of Morton-style factorial coding and illustrated how it may help analyze information integration and perceptual organization in the brain. We showed that by focusing on average responses one may miss the existence of factorial coding mechanisms that become only apparent when analyzing spike count histograms. The results of our study suggest that V1 represents color using a Morton-style factorial code. This may provide some cues to help understand perceptual coding in the brain and to develop new unsupervised learning algorithms. While methods like ICA (Bell & Sejnowski, 1997) develop independent codes, in Morton-style coding the goal is to make two or more external aspects of the world become independent when conditioning on internal representations.

Morton-style coding is optimal when the statistics of stimulus and background exhibit a particular property: when conditioning on each possible response category (i.e., spike counts) the empirical likelihood ratios of stimulus and background factorize. Our study suggests that Morton coding of color in natural scenes should be optimal or approximately optimal, a prediction that can be tested via statistical analysis of color in natural scenes.

## Acknowledgments

This project was supported by NSF's grant ITR IIS-0223052.

# 5 References

Bell, A., & Sejnowski, T. (1997). The 'independent components' of natural scenes are edge filters. *Vision Research*, *37(23)*, 3327–3338.

Brown, R. O., & MacLeod, D. I. A. (1997). Color appearance depends on the variance of surround colors. *Current Biology*, (7), 844–849.

Derrington, A. M., Krauskopf, J., & Lennie, P. (1984). Chromatic mechanisms in lateral geniculate nucleus of macaque. *Journal of Physiology*, *357*, 241–265.

Domingos, P., & Pazzani, M. (1997). On the optimality of the simple Bayesian classifier under zero-one loss. *Journal of Machine Learning*, *29*, 103–130.

Massaro, D. W. (1987a). Categorical perception: A fuzzy logical model of categorization behavior. In S. Harnad (Ed.), *Categorical perception*. Cambridge,England: Cambridge University Press.

Massaro, D. W. (1987b). *Speech perception by ear and eye: A paradigm for psychological research*. Hillsdale, NJ: Erlbaum.

Massaro, D. W. (1989a). *Perceiving talking faces*. Cambridge, Massachusetts: MIT Press.

Massaro, D. W. (1989b). Testing between the TRACE model and the fuzzy logical model of speech perception. *Cognitive Psychology*, *21*, 398–421.

Morton, J. (1969). The interaction of information in word recognition. *Psychological Review*, *76*, 165–178.

Movellan, J. R., & McClelland, J. L. (2001). The Morton-Massaro law of information integration: Implications for models of perception. *Psychological Review*, (1), 113–148.

Stockman, A., MaCleod, D. I. A., & Johnson, N. E. (1993). Spectral sensitivities of the human cones. *Journal of the Optical Society of America A*, (10), 2491–2521.

Wachtler, T., Sejnowski, T. J., & Albright, T. D. (2003). Representation of color stimuli in awake macaque primary visual cortex. *Neuron*, *37*, 1–20.

Wesner, M. F., & Shevell, S. K. (1992). Color perception within a chromatic context: Changes in red/green equilibria caused by noncontiguous light. *Vision Research*, (32), 1623–1634.

Zeki, S. (1983). Colour coding in cerebral cortex: the responses of wavelength selective and colour-coded cells in monkey visual cortex to changes in wavelenght composition. *Neuroscience*, *9*, 767–781.
